# Onset-based Sound Segmentation

**Leslie S. Smith**
CCCN/Department of Computer Science
University of Stirling
Stirling FK9 4LA
Scotland

## Abstract

A technique for segmenting sounds using processing based on mammalian early auditory processing is presented. The technique is based on features in sound which neuron spike recording suggests are detected in the cochlear nucleus. The sound signal is bandpassed and each signal processed to enhance onsets and offsets. The onset and offset signals are compressed, then clustered both in time and across frequency channels using a network of integrate-and-fire neurons. Onsets and offsets are signalled by spikes, and the timing of these spikes used to segment the sound.

## 1 Background

Traditional speech interpretation techniques based on Fourier transforms, spectrum recoding, and a hidden Markov model or neural network interpretation stage have limitations both in continuous speech and in interpreting speech in the presence of noise, and this has led to interest in front ends modelling biological auditory systems for speech interpretation systems (Ainsworth and Meyer 92; Cosi 93; Cole et al 95).

Auditory modelling systems use similar early auditory processing to that used in biological systems. Mammalian auditory processing uses two ears, and the incoming signal is filtered first by the pinna (external ear) and the auditory canal before it causes the tympanic membrane (eardrum) to vibrate. This vibration is then passed on through the bones of the middle ear to the oval window on the cochlea. Inside the cochlea, the pressure wave causes a pattern of vibration to occur on the basilar membrane. This appears to be an active process using both the inner and outer hair cells of the organ of Corti. The movement is detected by the inner hair cells and turned into neural impulses by the neurons of the spiral ganglion. These pass down the auditory nerve, and arrive at various parts of the cochlear nucleus. From there, nerve fibres innervate other areas: the lateral and medial nuclei of the superior olive,

and the inferior colliculus, for example. (See (Pickles 88)).

Virtually all modern sound or speech interpretation systems use some form of band-pass filtering, following the biology as far as the cochlea. Most use Fourier transforms to perform a calculation of the energy in each band over some time period, usually between 25 and 75 ms. This is not what the cochlea does. Auditory modelling front ends differ in the extent and length to which they follow animal early auditory processing, but the term generally implies at least that wideband filters are used, and that high temporal resolution is maintained in the initial stages. This means the use of filtering techniques, rather than Fourier transforms in the bandpass stage. Such filtering systems have been implemented by Patterson and Holdsworth (Patterson and Holdsworth 90; Slaney 93), and placed directly in silicon (Lazzaro and Mead 89; Lazzaro et al 93; Liu et al 93; Fragniere and van Schaik 94).

Some auditory models have moved beyond cochlear filtering. The inner hair cell has been modelled by either simple rectification (Smith 94) or has been based on the work of (Meddis 88) for example (Patterson and Holdsworth 90; Cosi 93; Brown 92). Lazzaro has experimented with a silicon version of Licklider's autocorrelation processing (Licklider 51; Lazzaro and Mead 89). Others such as (Wu et al 1989; Blackwood et al 1990; Ainsworth and Meyer 92; Brown 92; Berthommier 93; Smith 94) have considered the early brainstem nuclei, and their possible contribution, based on the neurophysiology of the different cell types (Pickles 88; Blackburn and Sachs 1989; Kim et al 90).

Auditory model-based systems have yet to find their way into mainstream speech recognition systems (Cosi 93). The work presented here uses auditory modelling up to onset cells in the cochlear nucleus. It adds a temporal neural network to clean up the segmentation produced. This part has been filed as a patent (Smith 95). Though the system has some biological plausibility, the aim is an effective data-driven segmentation technique implementable in silicon.

## 2    Techniques used

Digitized sound was applied to an auditory front end, (Patterson and Holdsworth 90), which bandpassed the sound into channels each with bandwidth $24.7(4.37F_c + 1)$Hz, where $F_c$ is the centre frequency (in KHz) of the band (Moore and Glasberg 83). These were rectified, modelling the effect of the inner hair cells. The signals produced bear some resemblance to that in the auditory nerve. The real system has far more channels and each nerve channel carries spike-coded information. The coding here models the signal in a population of neighboring auditory nerve fibres.

### 2.1    The onset-offset filter

The signal present in the auditory nerve is stronger near the onset of a tone than later (Pickles 88). This effect is much more pronounced in certain cell types of the cochlear nucleus. These fire strongly just after the onset of a sound in the band to which they are sensitive, and are then silent. This emphasis on onsets was modelled by convolving the signal in each band with a filter which computes two averages, a more recent one, and a less recent one, and subtracts the less recent one from the more recent one. One biologically possible justification for this is to consider that a neuron is receiving the same driving input twice, one excitatorily, and the other inhibitorily; the excitatory input has a shorter time-constant than the inhibitory input. Both exponentially weighted averages, and averages formed using a Gaussian filter have been tried (Smith 94), but the former place too much emphasis on the most recent part of the signal, making the latter more effective.

The filter output for input signal $s(x)$ is

$$O(t, k, r) = \int_0^t (f(t - x, k) - f(t - x, k/r))s(x)dx \qquad (1)$$

where $f(x, y) = \sqrt{y} \exp(-yx^2)$. $k$ and $r$ determine the rise and fall times of the pulses of sound that the system is sensitive to. We used $k = 1000$, $r = 1.2$, so that the SD of the Gaussians are 24.49ms and 22.36ms. The convolving filter has a positive peak at 0, crosses 0 at 22.39ms. and is then negative. With these values, the system is sensitive to energy rises and falls which occur in the envelopes of everyday sounds. A positive onset-offset signal implies that the bandpassed signal is increasing in intensity, and a negative onset-offset signal implies that it is decreasing in intensity. The convolution used is a sound analog of the difference of Gaussians operator used to extract black/white and white/black edges in monochrome images (Marr and Hildreth 80). In (Smith 94) we performed sound segmentation directly on this signal.

## 2.2 Compressing the onset-offset signal

The onset-offset signal was divided into two positive-going signals, an onset signal consisting of the positive-going part, and an offset signal consisting of the inverted negative-going part. Both were compressed logarithmically (where $\log(x)$ was taken as 0 for $0 \leq x \leq 1$). This increases the dynamical range of the system, and models compressive biological effects. The compressed onset signal models the output of a population of onset cells. This technique for producing an onset signal is related to that of (Wu et al 1989; Cosi 93).

## 2.3 The integrate-and-fire neural network

To segment the sound using the onset and offset signals, they need to be integrated across frequency bands and across time. This temporal and tonotopic clustering was achieved using a network of integrate-and-fire units. An integrate-and-fire unit accumulates its weighted input over time. The activity of the unit $A$, is initially 0, and alters according to

$$\frac{dA}{dt} = I(t) - \gamma A \qquad (2)$$

where $I(t)$ is the input to the neuron and $\gamma$, the dissipation, describes the leakiness of the integration. When $A$ reaches a threshold, the unit fires (i.e. emits a pulse), and $A$ is reset to 0. After firing, there is a period of insensitivity to input, called the refractory period. Such neurons are discussed in, e.g. (Mirolla and Strogatz 90).

One integrate-and-fire neuron was used per channel: this neuron received input either from a single channel, or from a set of adjacent channels, all with equal positive weighting. The output of each neuron was fed back to a set of adjacent neurons, again with a fixed positive weight, one time step (here 0.5ms) later. Because of the leaky nature of the accumulation of activity, excitatory input to the neuron arriving when its activation is near threshold has a larger effect on the next firing time than excitatory input arriving when activation is lower. Thus, if similar input is applied to a set of neurons in adjacent channels, the effect of the inter-neuron connections is that when the first one fires, its neighbors fire almost immediately. This allows a network of such neurons to cluster the onset or offset signals, producing a sharp burst of spikes across a number of channels providing unambiguous onsets or offsets.

The external and internal weights of the network were adjusted so that onset or offset input alone allowed neurons to fire, while internal input alone was not enough

to cause firing. The refractory period used was set to 50ms for the onset system, and 5ms for the offset system. For the onset system, the effect was to produce sharp onset firing responses across adjacent channels in response to a sudden increase in energy in some channels, thus grouping onsets both tonotopically and temporally. This is appropriate for onsets, as these are generally brief and clearly marked. The output of this stage we call the onset map. Offsets tend to be more gradual. This is due to physical effects: for example, a percussive sound will start suddenly, as the vibrating element starts to move, but die away slowly as the vibration ceases (see (Gaver 93) for a discussion). Even when the vibration does stop suddenly, the sound will die away more slowly due to echoes. Thus we cannot reliably mark the offset of a sound: instead, we reduce the refractory period of the offset neurons, and produce a train of pulses marking the duration of the offset in this channel. We call the output of this stage the offset map.

## 3   Results

As the technique is entirely data-driven, it can be applied to sound from any source. It has been applied to both speech and musical sounds. Figure 1 shows the effect of applying the techniques discussed to a short piece of speech. Fig 1c shows that the neural network integrates the onset timings across the channels, allowing these onsets to be used for segmentation. The simplest technique is to divide up the continuous speech at each onset: however,to ensure that the occasional onset in a single channel does not confuse the system, and that onsets which occur near to each other do not result in very short segments we demanded that a segmentation boundary have at least 6 onsets inside a period of 10ms, and the minimum segment length was set to 25ms.

The utterance *Neural information processing systems* has phonetic representation:

$$/\text{njɨrlːənfərmeʃənprosɛsəŋsɪstəms}/$$

and is segmented into the following 19 segments:

/n/, jɨ/, /r/, /lə/, /ə/, /nf/, /ərm/, /e/, /ʃ/, /ən/, /pro/, /os/, /ɛs/, /əŋ/, /s/, /ɪ/, /st/, /əm/, /s/

The same text spoken more slowly (over 4.38s, rather than 2.31s) has phonetic representation:

$$/\text{njuralːənfɪrmeʃənprosɛsɪŋsɪstəms}/$$

Segmenting using this technique gives the following 25 segments:

/n/, /ju/, /u/, /r/, /a/, /al/, /l/, //, /ən/, /f/, /ɪrm/, /e/, /ʃ/, /ən/, /nː/, /pr/, /ro/, /os/, /ɛs/, /ɪŋ/, /s/, /ɪ/, /st/, /əm/, /s/

Although some phonemes are broken between segments, the system provides effective segmentation, and is relatively insensitive to speech rate. The system is also effective at finding speech inside certain types of noise (such as motor-bike noise), as can be seen in fig 1e and f.

The system has been used to segment sound from single musical instruments. Where these have clear breaks between notes this is straightforward: in (Smith 94) correct segmentation was achieved directly from the onset-offset signal but was not achieved for slurred sounds, in which the notes change smoothly. As is visible in figure 2c, the onsets here are clear using the network, and the segmentation produced is near-

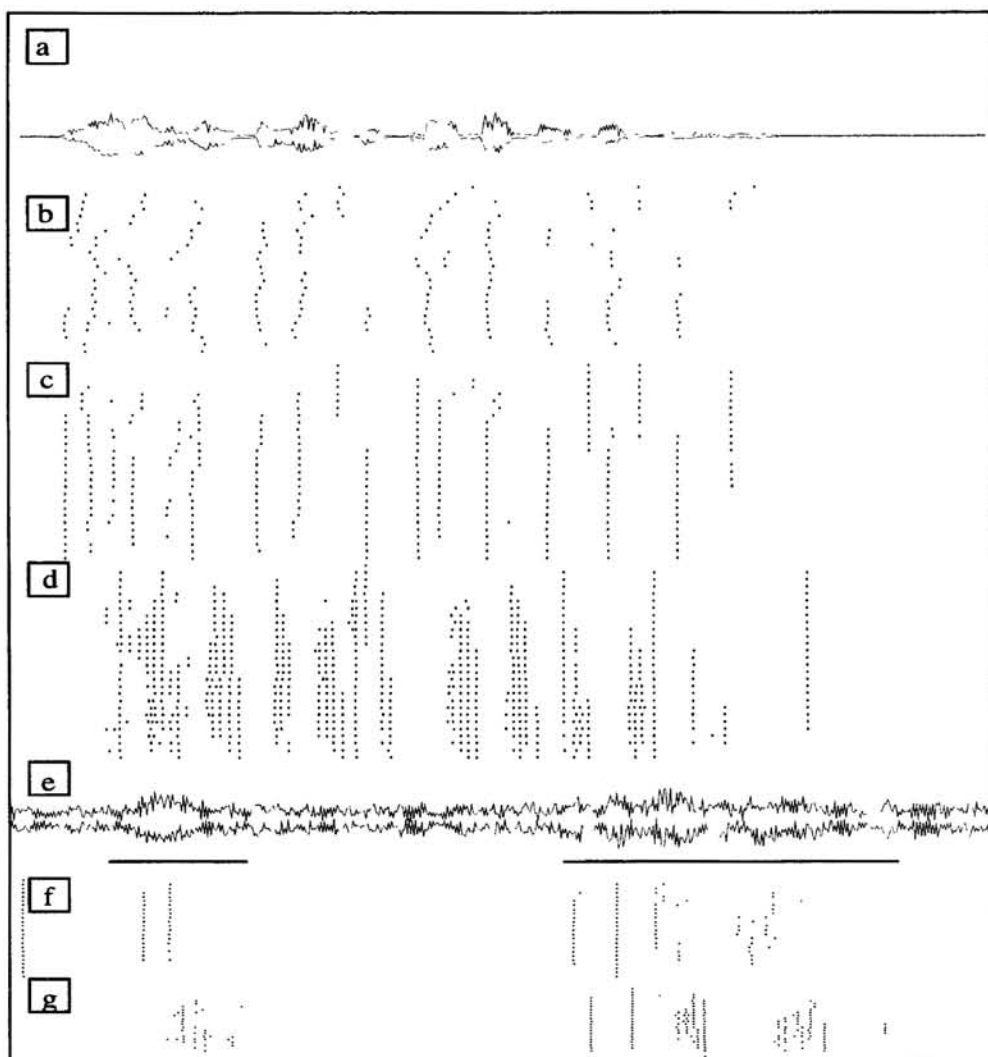

Figure 1: (a-d):Onset and Offset maps from author saying *Neural information processing systems* rapidly. a: envelope of original sound. b: onset map, from 28 channels, from 100Hz-6KHz. Onset filter parameters as in text; one neuron per channel, with no interconnection. Neuron refractory period is 50ms. c: as b, but network has input applied to 6 adjacent channels, and internal feedback to 10 channels. d: offset map produced similarly, with refractory period 5ms. e: envelope of *say, that's a nice bike* with motorbike noise in background (lines mark utterance). f, g: onset, offset maps for e.

perfect. Best results were obtained here when the input to the network is not spread across channels.

## 4   Conclusions and further work

An effective data driven segmentation technique based on onset feature detection and using integrate-and-fire neurons has been demonstrated. The system is relatively immune to broadband noise. Segmentation is not an end in itself: the effectiveness of any technique will depend on the eventual application.

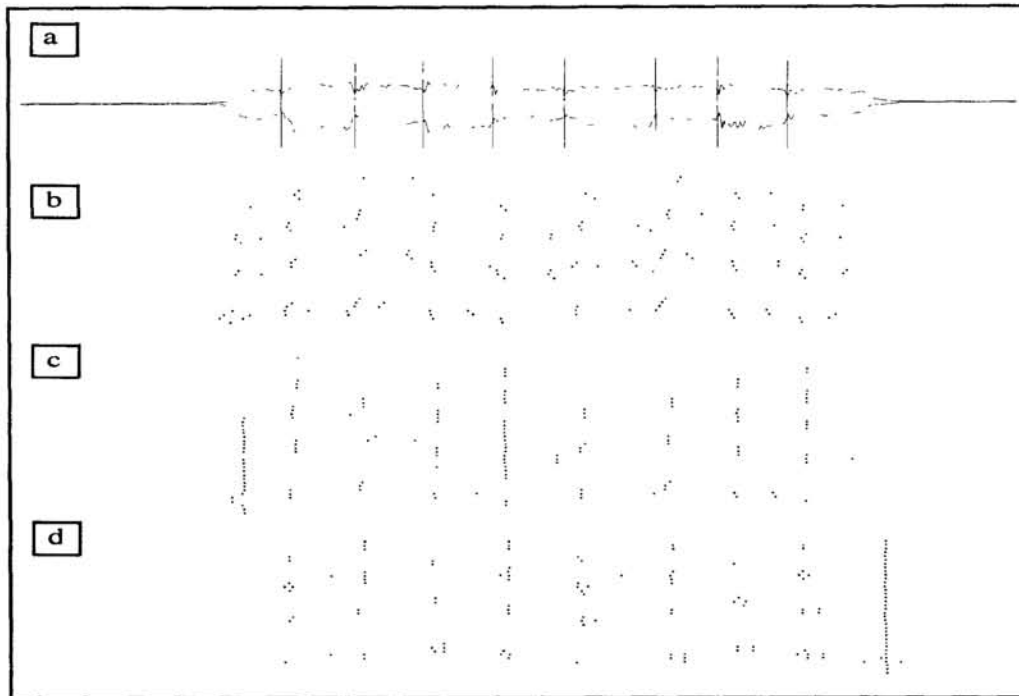

Figure 2: a: slurred flute sound, with vertical lines showing boundary between notes. b: onsets found using a single neuron per channel, and no interconnection. c: as b, but with internal feedback from each channel to 16 adjacent channels d: offsets found with refractory period 5ms.

The segmentation is currently not using the information on which bands the onsets occur in. We propose to extend this work by combining the segmentation described here with work streaming bands sharing same-frequency amplitude modulation. The aim of this is to extract sound segments from some subset of the bands, allowing segmentation and streaming to run concurrently.

## Acknowledgements

Many thanks are due to the members of the Centre for Cognitive and Computational Neuroscience at the University of Stirling.

## References

Ainsworth W. Meyer G. Speech analysis by means of a physiologically-based model of the cochlear nerve and cochlear nucleus. in *Visual representations of speech signals*. Cooke M. Beet S. eds. 1992.

Berthommier F.. Modelling neural responses of the intermediate auditory system, in *Mathematics applied to biology and medicine*, Demongeot J, Capasso V, Wuertz Publishing, Canada, 1993.

Blackburn C.C., Sachs M.B. Classification of unit types in the anteroventral cochlear nucleus: PST histograms and regularity analysis. . *J. Neurophysiology*, 62, 6, 1989.

Blackwood N., Meyer G., Aimsworth W. A Model of the processing of voiced plosives in the auditory nerve and cochlear nucleus, *Proceedings Inst of Acoustics*, 12, 10, 1990.

Brown G. *Computational Auditory Scene Analysis*, TR CS-92-22, Department of Computing Science, University of Sheffield, England, 1992.

Cole R., et al, The challenge of spoken language systems: research directions of the 90's, *IEEE Trans Speech and Audio Processing*, 3, 1, 1995.

Cosi P. On the use of auditory models in speech technology, in *Intelligent Perceptual Models*, LNCS 745, Springer Verlag, 1993.

Fragniere E., van Schaik A., Linear predictive coding of the speech signal using an analog cochlear model, MANTRA Internal Report, 94/2, MANTRA Center for Neuro-mimetic systems, EPFL, Lausanne, Switzerland, 1994.

Gaver W.W. What in the world do we hear?: an ecological approach to auditory event perception, *Ecological Psychology*, 5(1), 1-29, 1993.

Kim D.O. ,Sirianni J.G., Chang S.O., Responses of DCN-PVCN neurons and auditory nerve fibres in unanesthetized decerebrate cats to AM and pure tones: analysis with autocorrelation/power-spectrum, *Hearing Research*, 45, 95-113, 1990.

Lazzaro J., Mead C., Silicon modelling of pitch perception, *Proc Natl. Acad Sciences*, USA, 86, 9597-9601, 1989.

Lazzaro J., Wawrzynek J., Mahowald M., Sivilotti M., Gillespie D., Silicon auditory processors as computer peripherals, *IEEE Trans on Neural Networks*, 4, 3, May 1993.

Licklider J.C.R., A Duplex theory of pitch perception, *Experentia*, 7, 128-133, 1951.

Liu W., Andreou A.G., Goldstein M.H., Analog cochlear model for multiresolution speech analysis, *Advances in Neural Information Processing Systems 5*, Hanson S.J., Cowan J.D., Lee Giles C. (eds), Morgan Kaufmann, 1993.

Marr D., Hildreth E. Theory of edge detection, *Proc. Royal Society of London B*, 207, 187-217, 1980.

Meddis R., Simulation of auditory-neural transduction: further studies, *J. Acoust Soc Am.* 83, 3, 1988.

Moore B.C.J., Glasberg B.R. Suggested formulae for calculating auditory-filter bandwidths and excitation patterns, *J Acoust Soc America*, 74, 3, 1983.

Mirollo R.E., Strogatz S.H. Synchronization of pulse-coupled biological oscillators, *SIAM J. Appl Math*, 50, 6, 1990.

Patterson R., Holdsworth J. (1990). *An Introduction to Auditory Sensation Processing*, in AAM HAP, Vol 1, No 1.

Pickles J.O. (1988). *An Introduction to the Physiology of Hearing*, 2nd Edition, Academic Press.

Slaney M., An efficient implementation of the Patterson-Holdsworth auditory filter bank, Apple technical report No 35, Apple Computer Inc, 1993.

Smith L.S. Sound segmentation using onsets and offsets, *J of New Music Research*, 23, 1, 1994.

Smith L.S. *Onset/offset coding for interpretation and segmentation of sound*, UK patent no 9505956.4, March 1995.

Wu Z.L., Schwartz J.L., Escudier P. A theoretical study of neural mechanisms specialized in the detection of articulatory-acoustic events, *Proc Eurospeech 89*, ed Tubach J.P., Mariani J.J., Paris, 1989.